# Learning syntactic patterns for automatic hypernym discovery

**Rion Snow**
Computer Science Department
Stanford University
Stanford, CA 94305
rion@cs.stanford.edu

**Daniel Jurafsky**
Linguistics Department
Stanford University
Stanford, CA 94305
jurafsky@stanford.edu

**Andrew Y. Ng**
Computer Science Department
Stanford University
Stanford, CA 94305
ang@cs.stanford.edu

## Abstract

Semantic taxonomies such as WordNet provide a rich source of knowledge for natural language processing applications, but are expensive to build, maintain, and extend. Motivated by the problem of automatically constructing and extending such taxonomies, in this paper we present a new algorithm for automatically learning hypernym (is-a) relations from text. Our method generalizes earlier work that had relied on using small numbers of hand-crafted regular expression patterns to identify hypernym pairs. Using "dependency path" features extracted from parse trees, we introduce a general-purpose formalization and generalization of these patterns. Given a training set of text containing known hypernym pairs, our algorithm automatically extracts useful dependency paths and applies them to new corpora to identify novel pairs. On our evaluation task (determining whether two nouns in a news article participate in a hypernym relationship), our automatically extracted database of hypernyms attains both higher precision and higher recall than WordNet.

## 1   Introduction

Semantic taxonomies and thesauri such as WordNet [5] are a key source of knowledge for natural language processing applications, and provide structured information about semantic relations between words. Building such taxonomies, however, is an extremely slow and labor-intensive process. Further, semantic taxonomies are invariably limited in scope and domain, and the high cost of extending or customizing them for an application has often limited their usefulness. Consequently, there has been significant recent interest in finding methods for automatically learning taxonomic relations and constructing semantic hierarchies. [1, 2, 3, 4, 6, 8, 9, 13, 15, 17, 18, 19, 20, 21]

In this paper, we build an automatic classifier for the *hypernym/hyponym* relation. A noun X is a hyponym of a noun Y if X is a subtype or instance of Y. Thus "Shakespeare" is a hyponym of "author" (and conversely "author" is a hypernym of "Shakespeare"), "dog" is a hyponym of "canine", "desk" is a hyponym of "furniture", and so on.

Much of the previous work on automatic semantic classification of words has been based on a key insight first articulated by Hearst [8], that the presence of certain "lexico-syntactic patterns" can indicate a particular semantic relationship between two nouns. Hearst noticed that, for example, linking two noun phrases (NPs) via the constructions "Such $NP_Y$ as $NP_X$", or "$NP_X$ and other $NP_Y$", often implies that $NP_X$ is a hyponym of $NP_Y$, i.e., that $NP_X$ is a kind of $NP_Y$. Since then, several researchers have used a small number (typically less than ten) of hand-crafted patterns like these to try to automatically label such semantic

relations [1, 2, 6, 13, 17, 18]. While these patterns have been successful at identifying some examples of relationships like hypernymy, this method of lexicon construction is tedious and severely limited by the small number of patterns typically employed.

Our goal is to use machine learning to automatically replace this hand-built knowledge. We first use examples of known hypernym pairs to automatically identify large numbers of useful lexico-syntactic patterns, and then combine these patterns using a supervised learning algorithm to obtain a high accuracy hypernym classifier. More precisely, our approach is as follows:

1. Training:

   (a) Collect noun pairs from corpora, identifying examples of hypernym pairs (pairs of nouns in a hypernym/hyponym relation) using WordNet.
   (b) For each noun pair, collect sentences in which both nouns occur.
   (c) Parse the sentences, and automatically extract patterns from the parse tree.
   (d) Train a hypernym classifier based on these features.

2. Test:

   (a) Given a pair of nouns in the test set, extract features and use the classifier to determine if the noun pair is in the hypernym/hyponym relation or not.

The rest of the paper is structured as follows. Section 2 introduces our method for automatically discovering patterns indicative of hypernymy. Section 3 then describes the setup of our experiments. In Section 4 we analyze our feature space, and in Section 5 we describe a classifier using these features that achieves high accuracy on the task of hypernym identification. Section 6 shows how this classifier can be improved by adding a new source of knowledge, coordinate terms.

## 2   Representing lexico-syntactic patterns with dependency paths

The first goal of our work is to automatically identify lexico-syntactic patterns indicative of hypernymy. In order to do this, we need a representation space for expressing these patterns. We propose the use of *dependency paths* as a general-purpose formalization of the space of lexico-syntactic patterns. Dependency paths have been used successfully in the past to represent lexico-syntactic relations suitable for semantic processing [11].

A dependency parser produces a dependency tree that represents the syntactic relations between words by a list of edge tuples of the form:
$(word_1,\text{CATEGORY}_1\text{:RELATION:CATEGORY}_2, word_2)$. In this formulation each $word$ is the stemmed form of the word or multi-word phrase (so that "*authors*" becomes "*author*"), and corresponds to a specific node in the dependency tree; each $category$ is the part of speech label of the corresponding word (e.g., N for noun or PREP for preposition); and the $relation$ is the directed syntactic relationship exhibited between $word_1$ and $word_2$ (e.g., OBJ for object, MOD for modifier, or CONJ for conjunction), and corresponds to a specific link in the tree. We may then define our space of lexico-syntactic patterns to be all shortest paths of four links or less between any two nouns in a dependency tree. Figure 1 shows the partial dependency tree for the sentence fragment *"...such authors as Herrick and Shakespeare"* generated by the broad-coverage dependency parser MINIPAR [10].

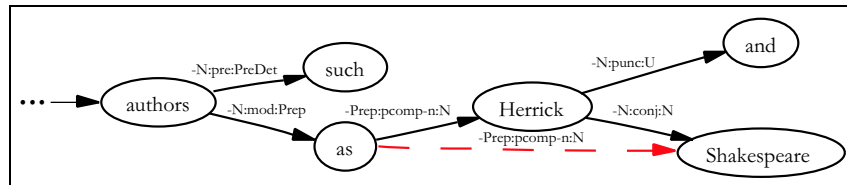

Figure 1: MINIPAR dependency tree example with transform

| | |
|---|---|
| $NP_X$ and other $NP_Y$: | (*and*,U:PUNC:N),-N:CONJ:N, (*other*,A:MOD:N) |
| $NP_X$ or other $NP_Y$: | (*or*,U:PUNC:N),-N:CONJ:N, (*other*,A:MOD:N) |
| $NP_Y$ such as $NP_X$: | N:PCOMP-N:PREP,*such_as,such_as*,PREP:MOD:N |
| Such $NP_Y$ as $NP_X$: | N:PCOMP-N:PREP,*as,as*,PREP:MOD:N,(*such*,PREDET:PRE:N) |
| $NP_Y$ including $NP_X$: | N:OBJ:V,*include,include*,V:I:C,*dummy_node,dummy_node*,C:REL:N |
| $NP_Y$, especially $NP_X$: | -N:APPO:N,(*especially*,A:APPO-MOD:N) |

Table 1: Dependency path representations of Hearst's patterns

We then remove the original nouns in the noun pair to create a more general pattern. Each dependency path may then be presented as an ordered list of dependency tuples. We extend this basic MINIPAR representation in two ways: first, we wish to capture the fact that certain function words like "such" (in "such NP as NP") or "other" (in "NP and other NP") are important parts of lexico-syntactic patterns. We implement this by adding optional "satellite links" to each shortest path, i.e., single links not already contained in the dependency path added on either side of each noun. Second, we capitalize on the distributive nature of the syntactic *conjunction* relation (nouns linked by "and" or "or", or in comma-separated lists) by distributing dependency links across such conjunctions. For example, in the simple 2-member conjunction chain of "Herrick" and "Shakespeare" in Figure 1, we add the entrance link "*as*, -PREP:PCOMP-N:N" to the single element "Shakespeare" (as a dotted line in the figure). Our extended dependency notation is able to capture the power of the hand-engineered patterns described in the literature. Table 1 shows the six patterns used in [1, 2, 8] and their corresponding dependency path formalizations.

## 3   Experimental paradigm

Our goal is to build a classifier which, when given an ordered pair of nouns, makes the binary decision of whether the nouns are related by hypernymy.

All of our experiments are based on a corpus of over 6 million newswire sentences.[1] We first parsed each of the sentences in the corpus using MINIPAR. We extract every pair of nouns from each sentence.

752,311 of the resulting unique noun pairs were labeled as Known Hypernym or Known Non-Hypernym using WordNet.[2] A noun pair $(n_i, n_j)$ is labeled Known Hypernym if $n_j$ is an ancestor of the first sense of $n_i$ in the WordNet hypernym taxonomy, and if the only "frequently-used"[3] sense of each noun is the first noun sense listed in WordNet. Note that $n_j$ is considered a hypernym of $n_i$ regardless of how much higher in the hierarchy it is with respect to $n_i$. A noun pair may be assigned to the second set of Known Non-Hypernym pairs if both nouns are contained within WordNet, but neither noun is an ancestor of the other in the WordNet hypernym taxonomy for any senses of either noun. Of our collected noun pairs, 14,387 were Known Hypernym pairs, and we assign the 737,924 most frequently occurring Known Non-Hypernym pairs to the second set; this number is selected to preserve the roughly 1:50 ratio of hypernym-to-non-hypernym pairs observed in our hand-labeled test set (discussed below).

We evaluated our binary classifiers in two ways. For both sets of evaluations, our classifier was given a pair of nouns from an unseen sentence and had to make a hypernym vs. non-hypernym decision. In the first style of evaluation, we compared the performance of our classifiers against the Known Hypernym versus Known Non-Hypernym labels assigned by

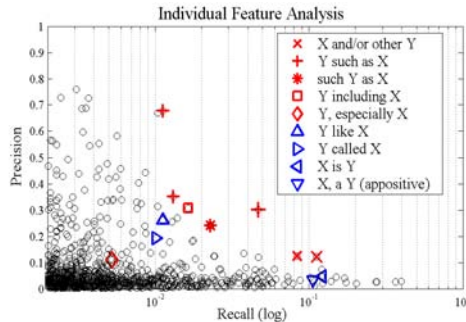
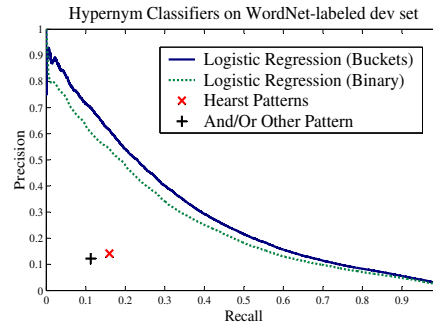

Figure 2: Hypernym pre/re for all features   Figure 3: Hypernym classifiers

WordNet. This provides a metric for how well our classifiers do at "recreating" WordNet's judgments.

For the second set of evaluations we hand-labeled a test set of 5,387 noun pairs from randomly-selected paragraphs within our corpus (with part-of-speech labels assigned by MINIPAR). The annotators were instructed to label each ordered noun pair as one of "hyponym-to-hypernym", "hypernym-to-hyponym", "coordinate", or "unrelated" (the coordinate relation will be defined in Section 6). As expected, the vast majority of pairs (5,122) were found to be unrelated by these measures; the rest were split evenly between hypernym and coordinate pairs (134 and 131, resp.).

Interannotator agreement was obtained between four labelers (all native speakers of English) on a set of 511 noun pairs, and determined for each task according to the averaged F-Score across all pairs of the four labelers. Agreement was 83% and 64% for the hypernym and coordinate term classification tasks, respectively.

## 4   Features: pattern discovery

Our first study focused on discovering which dependency paths might prove useful features for our classifiers. We created a feature lexicon of 69,592 dependency paths, consisting of every dependency path that occurred between at least five unique noun pairs in our corpus. To evaluate these features, we constructed a binary classifier for each pattern, which simply classifies a noun pair as hypernym/hyponym if and only if the specific pattern occurs at least once for that noun pair. Figure 2 depicts the precision and recall of all such classifiers (with recall at least .0015) on the WordNet-labeled data set.[4] Using this formalism we have been able to capture a wide variety of repeatable patterns between hypernym/hyponym noun pairs; in particular, we have been able to rediscover the hand-designed patterns originally proposed in [8] (the first five features, marked in red)[5], in addition to a number of new patterns not previously discussed (of which four are marked as blue triangles in Figure 2 and listed in Table 2. This analysis gives a quantitative justification to Hearst's initial intuition as to the power of hand-selected patterns; nearly all of Hearst's patterns are at the high-performance boundary of precision and recall for individual features.

| | |
|---|---|
| $NP_Y$ like $NP_X$: | N:PCOMP-N:PREP,*like,like*,PREP:MOD:N |
| $NP_Y$ called $NP_X$: | N:DESC:V,*call,call*,V:VREL:N |
| $NP_X$ is a $NP_Y$: | N:S:VBE,*be,be*,-VBE:PRED:N |
| $NP_X$, a $NP_Y$ (appositive): | N:APPO:N |

Table 2: Dependency path representations of other high-scoring patterns

| | |
|---|---|
| Best Logistic Regression (Buckets): | 0.3480 |
| Best Logistic Regression (Binary): | 0.3200 |
| Best Multinomial Naive Bayes: | 0.3175 |
| Best Complement Naive Bayes: | 0.3024 |
| Hearst Patterns: | 0.1500 |
| "And/Or Other" Pattern: | 0.1170 |

Table 3: Average maximum F-scores for cross validation on WordNet-labeled training set

## 5 A hypernym-only classifier

Our first hypernym classifier is based on the intuition that unseen noun pairs are more likely to be a hypernym pair if they occur in the test set with one or more lexico-syntactic patterns found to be indicative of hypernymy. We record in our noun pair lexicon each noun pair that occurs with at least five unique paths from our feature lexicon discussed in the previous section. We then create a feature count vector for each such noun pair. Each entry of the 69,592-dimension vector represents a particular dependency path, and contains the total number of times that that path was the shortest path connecting that noun pair in some dependency tree in our corpus. We thus define as our task the binary classification of a noun pair as a hypernym pair based on its feature vector of dependency paths.

We use the WordNet-labeled Known Hypernym / Known Non-Hypernym training set defined in Section 3. We train a number of classifiers on this data set, including multinomial Naive Bayes, complement Naive Bayes [16], and logistic regression. We perform model selection using 10-fold cross validation on this training set, evaluating each model based on its maximum F-Score averaged across all folds. The summary of average maximum F-scores is presented in Table 3, and the precision/recall plot of our best models is presented in Figure 3. For comparison, we evaluate two simple classifiers based on past work using only a handful of hand-engineered features; the first simply detects the presence of at least one of Hearst's patterns, arguably the previous best classifier consisting only of lexico-syntactic patterns, and as implemented for hypernym discovery in [2]. The second classifier consists of only the "NP and/or other NP" subset of Hearst's patterns, as used in the automatic construction of a noun-labeled hypernym taxonomy in [1]. In our tests we found greatest performance from a binary logistic regression model with 14 redundant threshold buckets spaced at the exponentially increasing intervals $\{1, 2, 4, ...4096, 8192\}$; our resulting feature space consists of 974,288 distinct binary features. These buckets are defined such that a feature corresponding to pattern $p$ at threshold $t$ will be activated by a noun pair $n$ if and only if $p$ has been observed to occur as a shortest dependency path between $n$ at least $t$ times.

Our classifier shows a dramatic improvement over previous classifiers; in particular, using our best logistic regression classifier trained on newswire corpora, we observe a 132% relative improvement of average maximum F-score over the classifier based on Hearst's patterns.

## 6 Using coordinate terms to improve hypernym classification

While our hypernym-only classifier performed better than previous classifiers based on hand-built patterns, there is still much room for improvement. As [2] points out, one problem with pattern-based hypernym classifiers in general is that within-sentence hypernym pattern information is quite sparse. Patterns are useful only for classifying noun pairs which happen to occur in the same sentence; many hypernym/hyponym pairs may simply not occur in the same sentence in the corpus. For this reason [2], following [1] suggests relying on a second source of knowledge: "coordinate" relations between nouns. The WordNet glossary defines *coordinate terms* as "nouns or verbs that have the same hypernym". Here we treat the coordinate relation as a symmetric relation that exists between two nouns that share at least one common ancestor in the hypernym taxonomy, and are therefore "the same kind of thing" at some level. Many methods exist for inferring that two nouns are coordinate terms (a common subtask in automatic thesaurus induction). We expect that

| | |
|---|---|
| Interannotator Agreement: | 0.6405 |
| Distributional Similarity Vector Space Model: | 0.3327 |
| Thresholded Conjunction Pattern Classifier: | 0.2857 |
| Best WordNet Classifier: | 0.2630 |

Table 4: Summary of maximum F-scores on hand-labeled coordinate pairs

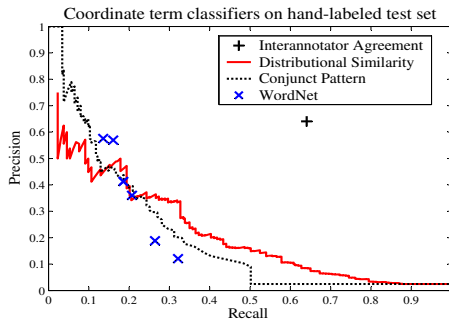
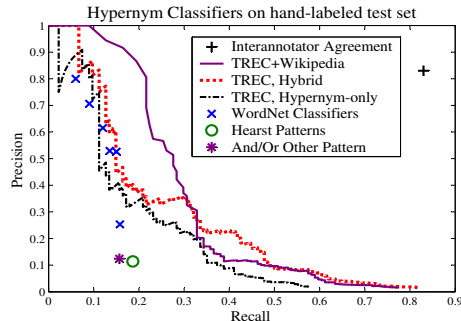

Figure 4: Coordinate classifiers on hand-labeled test set

Figure 5: Hypernym classifiers on hand-labeled test set

using coordinate information will increase the recall of our hypernym classifier: if we are confident that two nouns $n_i$, $n_j$ are coordinate terms, and that $n_j$ is a hyponym of $n_k$, we may then infer with higher probability that $n_i$ is similarly a hyponym of $n_k$—despite never having encountered the pair $(n_i, n_k)$ within a single sentence.

### 6.1 Coordinate Term Classification

Prior work for identifying coordinate terms includes automatic word sense clustering methods based on *distributional similarity* (e.g., [12, 14]) or on pattern-based techniques, specifically using the coordination pattern "X, Y, and Z" (e.g., [2]). We construct both types of classifier. First we construct a vector-space model similar to [12] using single MINIPAR dependency links as our distributional features.[6] We use the normalized similarity score from this model for coordinate term classification. We evaluate this classifier on our hand-labeled test set, where of 5,387 total pairs, 131 are labeled as "coordinate". For purposes of comparison we construct a series of classifiers from WordNet, which make the binary decision of determining whether two nouns are coordinate according to whether they share a common ancestor within $k$ nouns higher up in the hypernym taxonomy, for all $k$ from 1 to 6. Also, we compare a simple pattern-based classifier based on the conjunction pattern, which thresholds simply on the number of conjunction patterns found between a given pair. Results of this experiment are shown in Table 4 and Figure 4.

The strong performance of the simple conjunction pattern model suggests that it may be worth pursuing an extended pattern-based coordinate classifier along the lines of our hypernym classifier; for now, we proceed with our distributional similarity vector space model (with a 16% relative F-score improvement over the conjunction model) in the construction of a combined hypernym-coordinate hybrid classifier.

### 6.2 Hybrid hypernym-coordinate classification

We now combine our hypernym and coordinate models in order to improve hypernym classification. We define two probabilities of pair relationships between nouns: $P(n_i \underset{H}{<} n_j)$,

| | |
|---|---|
| Interannotator Agreement: | 0.8318 |
| TREC+Wikipedia Hypernym-only Classifier (Logistic Regression): | 0.3592 |
| TREC Hybrid Linear Interpolation Hypernym/Coordinate Model: | 0.3268 |
| TREC Hypernym-only Classifier (Logistic Regression): | 0.2714 |
| Best WordNet Classifier: | 0.2339 |
| Hearst Patterns Classifier: | 0.1417 |
| "And/Or Other" Pattern Classifier: | 0.1386 |

Table 5: Maximum F-Score of hypernym classifiers on hand-labeled test set

representing the probability that noun $n_i$ has $n_j$ as an ancestor in its hypernym hierarchy, and $P(n_i \underset{C}{\sim} n_j)$, the probability that nouns $n_i$ and $n_j$ are *coordinate terms*, i.e., that they share a common hypernym ancestor at some level. Defining the probability produced by our best hypernym-only classifier as $P_{old}(n_i \underset{H}{<} n_j)$, and a probability obtained by normalizing the similarity score from our coordinate classifier as $P(n_i \underset{C}{\sim} n_j)$, we apply a simple linear interpolation scheme to compute a new hypernymy probability. Specifically, for each pair of nouns $(n_i, n_k)$, we recompute the probability that $n_k$ is a hypernym of $n_i$ as:[7]

$$P_{new}(n_i \underset{H}{<} n_k) \propto \lambda_1 P_{old}(n_i \underset{H}{<} n_k) + \lambda_2 \sum_j P(n_i \underset{C}{\sim} n_j) P_{old}(n_j \underset{H}{<} n_k)$$

## 7 Results

Our hand-labeled dataset allows us to compare our classifiers with WordNet and the previous feature-based methods, now using the human labels as ground truth. Figure 5 shows the performance of each method in a precision/recall plot. We evaluated several classifiers based on the WordNet hypernym taxonomy.[8] The best WordNet-based results are plotted in Figure 5. Our logistic regression hypernym-only model trained on the newswire corpora has a 16% relative F-score improvement over the best WordNet classifier, while the combined hypernym/coordinate model has a 40% relative F-score improvement. Our best-performing classifier is a hypernym-only model additionally trained on the Wikipedia corpus, with an expanded feature lexicon of 200,000 dependency paths; this classifier shows a 54% improvement over WordNet. In Table 5 we list the maximum F-scores of each method. In Table 6 we analyze the disagreements between the highest F-score WordNet classifier and our combined hypernym/coordinate classifier.[9]

## 8 Conclusions

Our experiments demonstrate that automatic methods can be competitive with WordNet for the identification of hypernym pairs in newswire corpora. In future work we will use the presented method to automatically generate flexible, statistically-grounded hypernym taxonomies directly from corpora. These taxonomies will be made publicly available to complement existing semantic resources.

| Type of Noun Pair | Count | Example Pair |
|---|---|---|
| NE: Person | 7 | "John F. Kennedy / president", "Marlin Fitzwater / spokesman" |
| NE: Place | 7 | "Diamond Bar / city", "France / place" |
| NE: Company | 2 | "American Can / company", "Simmons / company" |
| NE: Other | 1 | "Is Elvis Alive / book" |
| Not Named Entity: | 9 | "earthquake / disaster", "soybean / crop" |

Table 6: Analysis of improvements over WordNet

## Acknowledgments

We thank Kayur Patel, Mona Diab, Allison Buckley, and Todd Huffman for useful discussions and assistance annotating data. R. Snow is supported by an NDSEG Fellowship sponsored by the DOD and AFOSR. This work is also supported by the ARDA AQUAINT program, and by the Department of the Interior/DARPA under contract number NBCHD030010.

## Footnotes

[1]The corpus contains articles from the Associated Press, Wall Street Journal, and Los Angeles Times, drawn from the TIPSTER 1, 2, 3, and TREC 5 corpora [7]. Our most recent experiments (presented in Section 6) include articles from Wikipedia (a popular web encyclopedia), extracted with the help of Tero Karvinen's Tero-dump software.

[2]We access WordNet 2.0 via Jason Rennie's WordNet::QueryData interface.

[3]A noun sense is determined to be "frequently-used" if it occurs at least once in the sense-tagged Brown Corpus Semantic Concordance files (as reported in the `cntlist` file distributed as part of WordNet 2.0). This determination is made so as to reduce the number of false hypernym/hyponym classifications due to highly polysemous nouns (nouns which have multiple meanings).

[4] Redundant features consisting of an identical base path to an identified pattern but differing only by an additional "satellite link" are marked in Figure 2 by smaller versions of the same symbol.

[5] We mark the single generalized "*conjunction* other" pattern -N:CONJ:N, (*other*,A:MOD:N) to represent both of Hearst's original "and other" and "or other" patterns.

[6]We use the same 6 million MINIPAR-parsed sentences used in our hypernym training set. Our feature lexicon consists of the 30,000 most frequent noun-connected dependency edges. We construct feature count vectors for each of the most frequently occurring 163,198 individual nouns. As in [12] we normalize these feature counts with pointwise mutual information, and compute as our measure of similarity the cosine coefficient between these normalized vectors.

[7]We constrain our parameters $\lambda_1$, $\lambda_2$ such that $\lambda_1 + \lambda_2 = 1$; we set these parameters using 10-fold cross-validation on our hand-labeled test set. For our final evaluation we use $\lambda_1 = 0.7$.

[8]We tried all combinations of the following parameters: the maximum number of senses of a hyponym for which to find hypernyms, the maximum distance between the hyponym and its hypernym in the WordNet taxonomy, and whether or not to allow synonyms. The WordNet model achieving the maximum F-score uses only the first sense of a hyponym and allows a maximum distance of 4 links between a hyponym and hypernym.

[9]There are 31 such disagreements, with WordNet agreeing with the human labels on 5 and our hybrid model agreeing on the other 26. We additionally inspect the types of noun pairs where our model improves upon WordNet, and find that at least 30% of our model's improvements are not restricted to Named Entities; given that the distribution of Named Entities among the labeled hypernyms in our test set is over 60%, this gives us hope that our classifier will perform well at the task of hypernym induction even in more general, non-newswire domains.

## References

[1] Caraballo, S.A. (2001) Automatic Acquisition of a Hypernym-Labeled Noun Hierarchy from Text. Brown University Ph.D. Thesis.

[2] Cederberg, S. & Widdows, D. (2003) Using LSA and Noun Coordination Information to Improve the Precision and Recall of Automatic Hyponymy Extraction. *Proc. of CoNLL-2003*, pp. 111–118.

[3] Ciaramita, M. & Johnson, M. (2003) Supersense Tagging of Unknown Nouns in WordNet. *Proc. of EMNLP-2003*.

[4] Ciaramita, M., Hofmann, T., & Johnson, M. (2003) Hierarchical Semantic Classification: Word Sense Disambiguation with World Knowledge. *Proc. of IJCAI-2003*.

[5] Fellbaum, C. (1998) WordNet: An Electronic Lexical Database. Cambridge, MA: MIT Press.

[6] Girju, R., Badulescu A., & Moldovan D. (2003) Learning Semantic Constraints for the Automatic Discovery of Part-Whole Relations. *Proc. of HLT-2003*.

[7] Harman, D. (1992) The DARPA TIPSTER project. *ACM SIGIR Forum* **26**(2), Fall, pp. 26–28.

[8] Hearst, M. (1992) Automatic Acquisition of Hyponyms from Large Text Corpora. *Proc. of the Fourteenth International Conference on Computational Linguistics, Nantes, France*.

[9] Hearst, M. & Schütze, H. (1993) Customizing a lexicon to better suit a computational task. In *Proc. of the ACL SIGLEX Workshop on Acquisition of Lexical Knowledge from Text*.

[10] Lin, D. (1998) Dependency-based Evaluation of MINIPAR. *Workshop on the Evaluation of Parsing Systems, Granada, Spain*

[11] Lin, D. & Pantel P. (2001) Discovery of Inference Rules for Question Answering. *Natural Language Engineering*, **7**(4), pp. 343–360.

[12] Pantel, P. (2003) Clustering by Committee. Ph.D. Dissertation. Department of Computing Science, University of Alberta.

[13] Pantel, P. & Ravichandran, D. (2004) Automatically Labeling Semantic Classes. *Proc. of NAACL-2004*.

[14] Pereira, F., Tishby, N., & Lee, L. (1993) Distributional Clustering of English Words. *Proc. of ACL-1993*, pp. 183–190.

[15] Ravichandran, D. & Hovy, E. (2002) Learning Surface Text Patterns for a Question Answering system. *Proc. of ACL-2002*.

[16] Rennie J., Shih, L., Teevan, J., & Karger, D. (2003) Tackling the Poor Assumptions of Naive Bayes Text Classifiers. *Proc. of ICLM-2003*.

[17] Riloff, E. & Shepherd, J. (1997) A Corpus-Based Approach for Building Semantic Lexicons. *Proc of EMNLP-1997*.

[18] Roark, B. & Charniak, E. (1998) Noun-phrase co-occurerence statistics for semi-automatic-semantic lexicon construction. *Proc. of ACL-1998*, 1110–1116.

[19] Tseng, H. (2003) Semantic classification of unknown words in Chinese. *Proc. of ACL-2003*.

[20] Turney, P.D., Littman, M.L., Bigham, J. & Shanyder, V. (2003) Combining independent modules to solve multiple-choice synonym and analogy problems. *Proc. of RANLP-2003*, pp. 482–489.

[21] Widdows, D. (2003) Unsupervised methods for developing taxonomies by combining syntactic and statistical information. *Proc. of HLT/NAACL 2003*, pp. 276–283.
